# Unified Inference for Variational Bayesian Linear Gaussian State-Space Models

**David Barber**
IDIAP Research Institute
rue du Simplon 4, Martigny, Switzerland
david.barber@idiap.ch

**Silvia Chiappa**
IDIAP Research Institute
rue du Simplon 4, Martigny, Switzerland
silvia.chiappa@idiap.ch

## Abstract

Linear Gaussian State-Space Models are widely used and a Bayesian treatment of parameters is therefore of considerable interest. The approximate Variational Bayesian method applied to these models is an attractive approach, used successfully in applications ranging from acoustics to bioinformatics. The most challenging aspect of implementing the method is in performing inference on the hidden state sequence of the model. We show how to convert the inference problem so that standard Kalman Filtering/Smoothing recursions from the literature may be applied. This is in contrast to previously published approaches based on Belief Propagation. Our framework both simplifies and unifies the inference problem, so that future applications may be more easily developed. We demonstrate the elegance of the approach on Bayesian temporal ICA, with an application to finding independent dynamical processes underlying noisy EEG signals.

## 1 Linear Gaussian State-Space Models

Linear Gaussian State-Space Models (LGSSMs)[1] are fundamental in time-series analysis [1, 2, 3]. In these models the observations $v_{1:T}$[2] are generated from an underlying dynamical system on $h_{1:T}$ according to:

$$v_t = Bh_t + \eta_t^v, \ \ \eta_t^v \sim \mathcal{N}(\mathbf{0}_V, \Sigma_V), \qquad h_t = Ah_{t-1} + \eta_t^h, \ \ \eta_t^h \sim \mathcal{N}(\mathbf{0}_H, \Sigma_H),$$

where $\mathcal{N}(\mu, \Sigma)$ denotes a Gaussian with mean $\mu$ and covariance $\Sigma$, and $\mathbf{0}_X$ denotes an $X$-dimensional zero vector. The observation $v_t$ has dimension $V$ and the hidden state $h_t$ has dimension $H$. Probabilistically, the LGSSM is defined by:

$$p(v_{1:T}, h_{1:T}|\Theta) = p(v_1|h_1)p(h_1)\prod_{t=2}^{T}p(v_t|h_t)p(h_t|h_{t-1}),$$

with $p(v_t|h_t) = \mathcal{N}(Bh_t, \Sigma_V)$, $p(h_t|h_{t-1}) = \mathcal{N}(Ah_{t-1}, \Sigma_H)$, $p(h_1) = \mathcal{N}(\mu, \Sigma)$ and where $\Theta = \{A, B, \Sigma_H, \Sigma_V, \mu, \Sigma\}$ denotes the model parameters. Because of the widespread use of these models, a Bayesian treatment of parameters is of considerable interest [4, 5, 6, 7, 8].

An exact implementation of the Bayesian LGSSM is formally intractable [8], and recently a Variational Bayesian (VB) approximation has been studied [4, 5, 6, 7, 9]. The most challenging part of implementing the VB method is performing inference over $h_{1:T}$, and previous authors have developed their own specialized routines, based on Belief Propagation, since standard LGSSM inference routines appear, at first sight, not to be applicable.

A key contribution of this paper is to show how the Variational Bayesian treatment of the LGSSM *can* be implemented using standard LGSSM inference routines. Based on the insight we provide, any standard inference method may be applied, including those specifically addressed to improve numerical stability [2, 10, 11]. In this article, we decided to describe the predictor-corrector and Rauch-Tung-Striebel recursions [2], and also suggest a small modification that reduces computational cost.

The Bayesian LGSSM is particularly of interest when strong prior constraints are needed to find adequate solutions. One such case is in EEG signal analysis, whereby we wish to extract sources that evolve independently through time. Since EEG is particularly noisy [12], a prior that encourages sources to have preferential dynamics is advantageous. This application is discussed in Section 4, and demonstrates the ease of applying our VB framework.

## 2 Bayesian Linear Gaussian State-Space Models

In the Bayesian treatment of the LGSSM, instead of considering the model parameters $\Theta$ as fixed, we define a prior distribution $p(\Theta|\hat{\Theta})$, where $\hat{\Theta}$ is a set of hyperparameters. Then:

$$p(v_{1:T}|\hat{\Theta}) = \int_{\Theta} p(v_{1:T}|\Theta)p(\Theta|\hat{\Theta}). \tag{1}$$

In a full Bayesian treatment we would define additional prior distributions over the hyperparameters $\hat{\Theta}$. Here we take instead the ML-II ('evidence') framework, in which the optimal set of hyperparameters is found by maximizing $p(v_{1:T}|\hat{\Theta})$ with respect to $\hat{\Theta}$ [6, 7, 9].

For the parameter priors, here we define Gaussians on the columns of $A$ and $B$[3]:

$$p(A|\alpha, \Sigma_H) \propto \prod_{j=1}^{H} e^{-\frac{\alpha_j}{2}(A_j - \hat{A}_j)^{\mathsf{T}} \Sigma_H^{-1}(A_j - \hat{A}_j)}, \qquad p(B|\beta, \Sigma_V) \propto \prod_{j=1}^{H} e^{-\frac{\beta_j}{2}(B_j - \hat{B}_j)^{\mathsf{T}} \Sigma_V^{-1}(B_j - \hat{B}_j)},$$

which has the effect of biasing the transition and emission matrices to desired forms $\hat{A}$ and $\hat{B}$. The conjugate priors for general inverse covariances $\Sigma_H^{-1}$ and $\Sigma_V^{-1}$ are Wishart distributions [7][4]. In the simpler case assumed here of diagonal covariances these become Gamma distributions [5, 7]. The hyperparameters are then $\hat{\Theta} = \{\alpha, \beta\}$[5].

### Variational Bayes

Optimizing Eq. (1) with respect to $\hat{\Theta}$ is difficult due to the intractability of the integrals. Instead, in VB, one considers the lower bound [6, 7, 9][6]:

$$\mathcal{L} = \log p(v_{1:T}|\hat{\Theta}) \geq H_q(\Theta, h_{1:T}) + \left\langle \log p(\Theta|\hat{\Theta}) \right\rangle_{q(\Theta)} + \left\langle E(h_{1:T}, \Theta) \right\rangle_{q(\Theta, h_{1:T})} \equiv \mathcal{F},$$

where

$$E(h_{1:T}, \Theta) \equiv \log p(v_{1:T}, h_{1:T}|\Theta).$$

$H_d(x)$ signifies the entropy of the distribution $d(x)$, and $\langle \cdot \rangle_{d(x)}$ denotes the expectation operator.

The key approximation in VB is $q(\Theta, h_{1:T}) \equiv q(\Theta)q(h_{1:T})$, from which one may show that, for optimality of $\mathcal{F}$,

$$q(h_{1:T}) \propto e^{\langle E(h_{1:T}, \Theta) \rangle_{q(\Theta)}}, \qquad q(\Theta) \propto p(\Theta|\hat{\Theta})e^{\langle E(h_{1:T}, \Theta) \rangle_{q(h_{1:T})}}.$$

These coupled equations need to be iterated to convergence. The updates for the parameters $q(\Theta)$ are straightforward and are given in Appendices A and B. Once converged, the hyperparameters are updated by maximizing $\mathcal{F}$ with respect to $\hat{\Theta}$, which lead to simple update formulae [7].

Our main concern is with the update for $q(h_{1:T})$, for which this paper makes a departure from treatments previously presented.

# 3 Unified Inference on $q(h_{1:T})$

Optimally $q(h_{1:T})$ is Gaussian since, up to a constant, $\langle E(h_{1:T}, \Theta) \rangle_{q(\Theta)}$ is quadratic in $h_{1:T}$[7]:

$$-\frac{1}{2}\sum_{t=1}^{T}\left[\left\langle (v_t - Bh_t)^{\mathsf{T}}\Sigma_V^{-1}(v_t - Bh_t)\right\rangle_{q(B,\Sigma_V)} + \left\langle (h_t - Ah_{t-1})^{\mathsf{T}}\Sigma_H^{-1}(h_t - Ah_{t-1})\right\rangle_{q(A,\Sigma_H)}\right].$$
(2)

In addition, optimally, $q(A|\Sigma_H)$ and $q(B|\Sigma_V)$ are Gaussians (see Appendix A), so we can easily carry out the averages in Eq. (2). The further averages over $q(\Sigma_H)$ and $q(\Sigma_V)$ are also easy due to conjugacy. Whilst this defines the distribution $q(h_{1:T})$, quantities such as $q(h_t)$, required for example for the parameter updates (see the Appendices), need to be inferred from this distribution. Clearly, in the non-Bayesian case, the averages over the parameters are not present, and the above simply represents the posterior distribution of an LGSSM whose visible variables have been clamped into their evidential states. In that case, inference can be performed using any standard LGSSM routine. Our aim, therefore, is to try to represent the *averaged* Eq. (2) directly as the posterior distribution $\tilde{q}(h_{1:T}|\tilde{v}_{1:T})$ of an LGSSM, for some suitable parameter settings.

**Mean + Fluctuation Decomposition**

A useful decomposition is to write

$$\left\langle (v_t - Bh_t)^{\mathsf{T}}\Sigma_V^{-1}(v_t - Bh_t)\right\rangle_{q(B,\Sigma_V)} = \underbrace{(v_t - \langle B\rangle h_t)^{\mathsf{T}}\langle\Sigma_V^{-1}\rangle(v_t - \langle B\rangle h_t)}_{mean} + \underbrace{h_t^{\mathsf{T}}S_B h_t}_{fluctuation},$$

and similarly

$$\left\langle (h_t - Ah_{t-1})^{\mathsf{T}}\Sigma_H^{-1}(h_t - Ah_{t-1})\right\rangle_{q(A,\Sigma_H)} = \underbrace{(h_t - \langle A\rangle h_{t-1})^{\mathsf{T}}\langle\Sigma_H^{-1}\rangle(h_t - \langle A\rangle h_{t-1})}_{mean} + \underbrace{h_{t-1}^{\mathsf{T}}S_A h_{t-1}}_{fluctuation},$$

where the parameter covariances are $S_B \equiv \langle B^{\mathsf{T}}\Sigma_V^{-1}B\rangle - \langle B\rangle^{\mathsf{T}}\langle\Sigma_V^{-1}\rangle\langle B\rangle = VH_B^{-1}$ and $S_A \equiv \langle A^{\mathsf{T}}\Sigma_H^{-1}A\rangle - \langle A\rangle^{\mathsf{T}}\langle\Sigma_H^{-1}\rangle\langle A\rangle = HH_A^{-1}$ (for $H_A$ and $H_B$ defined in Appendix A). The mean terms simply represent a clamped LGSSM with averaged parameters. However, the extra contributions from the fluctuations mean that Eq. (2) cannot be written as a clamped LGSSM with averaged parameters. In order to deal with these extra terms, our idea is to treat the fluctuations as arising from an augmented visible variable, for which Eq. (2) can then be considered as a clamped LGSSM.

**Inference Using an Augmented LGSSM**

To represent Eq. (2) as an LGSSM $\tilde{q}(h_{1:T}|\tilde{v}_{1:T})$, we may augment $v_t$ and $B$ as[8]:

$$\tilde{v}_t = vert(v_t, \mathbf{0}_H, \mathbf{0}_H), \qquad \tilde{B} = vert(\langle B\rangle, U_A, U_B),$$

where $U_A$ is the Cholesky decomposition of $S_A$, so that $U_A^{\mathsf{T}}U_A = S_A$. Similarly, $U_B$ is the Cholesky decomposition of $S_B$. The equivalent LGSSM $\tilde{q}(h_{1:T}|\tilde{v}_{1:T})$ is then completed by specifying[9]

$$\tilde{A} \equiv \langle A\rangle, \quad \tilde{\Sigma}_H \equiv \langle\Sigma_H^{-1}\rangle^{-1}, \quad \tilde{\Sigma}_V \equiv diag(\langle\Sigma_V^{-1}\rangle^{-1}, I_H, I_H), \quad \tilde{\mu} \equiv \mu, \quad \tilde{\Sigma} \equiv \Sigma.$$

The validity of this parameter assignment can be checked by showing that, up to negligible constants, the exponent of this augmented LGSSM has the same form as Eq. (2)[10]. Now that this has been written as an LGSSM $\tilde{q}(h_{1:T}|\tilde{v}_{1:T})$, standard inference routines in the literature may be applied to compute $q(h_t|v_{1:T}) = \tilde{q}(h_t|\tilde{v}_{1:T})$ [1, 2, 11][11].

**Algorithm 1** LGSSM: Forward and backward recursive updates. The smoothed posterior $p(h_t|v_{1:T})$ is returned in the mean $\hat{h}_t^T$ and covariance $P_t^T$.

---

**procedure** FORWARD
    1a: $P \leftarrow \Sigma$
    1b: $P \leftarrow D\Sigma$, where $D \equiv I - \Sigma U_{AB}\left(I + U_{AB}^{\mathsf{T}}\Sigma U_{AB}\right)^{-1} U_{AB}^{\mathsf{T}}$
    2a: $\hat{h}_1^0 \leftarrow \mu$
    2b: $\hat{h}_1^0 \leftarrow D\mu$
    3: $K \leftarrow PB^{\mathsf{T}}(BPB^{\mathsf{T}} + \Sigma_V)^{-1}$, $P_1^1 \leftarrow (I - KB)P$, $\hat{h}_1^1 \leftarrow \hat{h}_1^0 + K(v_t - B\hat{h}_1^0)$
    **for** $t \leftarrow 2, T$ **do**
        4: $P_t^{t-1} \leftarrow AP_{t-1}^{t-1}A^T + \Sigma_H$
        5a: $P \leftarrow P_t^{t-1}$
        5b: $P \leftarrow D_t P_t^{t-1}$, where $D_t \equiv I - P_t^{t-1}U_{AB}\left(I + U_{AB}^{\mathsf{T}}P_t^{t-1}U_{AB}\right)^{-1} U_{AB}^{\mathsf{T}}$
        6a: $\hat{h}_t^{t-1} \leftarrow A\hat{h}_{t-1}^{t-1}$
        6b: $\hat{h}_t^{t-1} \leftarrow D_t A\hat{h}_{t-1}^{t-1}$
        7: $K \leftarrow PB^{\mathsf{T}}(BPB^{\mathsf{T}} + \Sigma_V)^{-1}$, $P_t^t \leftarrow (I - KB)P$, $\hat{h}_t^t \leftarrow \hat{h}_t^{t-1} + K(v_t - B\hat{h}_t^{t-1})$
    **end for**
**end procedure**
**procedure** BACKWARD
    **for** $t \leftarrow T - 1, 1$ **do**
        $\overleftarrow{A_t} \leftarrow P_t^t A^{\mathsf{T}}(P_{t+1}^t)^{-1}$
        $P_t^T \leftarrow P_t^t + \overleftarrow{A_t}(P_{t+1}^T - P_{t+1}^t)\overleftarrow{A_t}^{\mathsf{T}}$
        $\hat{h}_t^T \leftarrow \hat{h}_t^t + \overleftarrow{A_t}(\hat{h}_{t+1}^T - A\hat{h}_t^t)$
    **end for**
**end procedure**

---

For completeness, we decided to describe the standard predictor-corrector form of the Kalman Filter, together with the Rauch-Tung-Striebel Smoother [2]. These are given in Algorithm 1, where $\tilde{q}(h_t|\tilde{v}_{1:T})$ is computed by calling the FORWARD and BACKWARD procedures.

We present two variants of the FORWARD pass. Either we may call procedure FORWARD in Algorithm 1 with parameters $\tilde{A}, \tilde{B}, \tilde{\Sigma}_H, \tilde{\Sigma}_V, \tilde{\mu}, \tilde{\Sigma}$ and the augmented visible variables $\tilde{v}_t$ in which we use steps 1a, 2a, 5a and 6a. This is exactly the predictor-corrector form of a Kalman Filter [2]. Otherwise, in order to reduce the computational cost, we may call procedure FORWARD with the parameters $\tilde{A}, \langle B \rangle, \tilde{\Sigma}_H, \left\langle \Sigma_V^{-1} \right\rangle^{-1}, \tilde{\mu}, \tilde{\Sigma}$ and the original visible variable $v_t$ in which we use steps 1b (where $U_{AB}^{\mathsf{T}}U_{AB} \equiv S_A + S_B$), 2b, 5b and 6b. The two algorithms are mathematically equivalent. Computing $q(h_t|v_{1:T}) = \tilde{q}(h_t|\tilde{v}_{1:T})$ is then completed by calling the common BACKWARD pass.

The important point here is that the reader may supply any standard Kalman Filtering/Smoothing routine, and simply call it with the appropriate parameters. In some parameter regimes, or in very long time-series, numerical stability may be a serious concern, for which several stabilized algorithms have been developed over the years, for example the square-root forms [2, 10, 11]. By converting the problem to a standard form, we have therefore unified and simplified inference, so that future applications may be more readily developed[12].

### 3.1 Relation to Previous Approaches

An alternative approach to the one above, and taken in [5, 7], is to write the posterior as

$$\log q(h_{1:T}) = \sum_{t=2}^{T} \phi_t(h_{t-1}, h_t) + const.$$

for suitably defined quadratic forms $\phi_t(h_{t-1}, h_t)$. Here the potentials $\phi_t(h_{t-1}, h_t)$ encode the averaging over the parameters $A, B, \Sigma_H, \Sigma_V$. The approach taken in [7] is to recognize this as a

pairwise Markov chain, for which the Belief Propagation recursions may be applied. The approach in [5] is based on a Kullback-Leibler minimization of the posterior with a chain structure, which is algorithmically equivalent to Belief Propagation. Whilst mathematically valid procedures, the resulting algorithms do not correspond to any of the standard forms in the Kalman Filtering/Smoothing literature, whose properties have been well studied [14].

## 4 An Application to Bayesian ICA

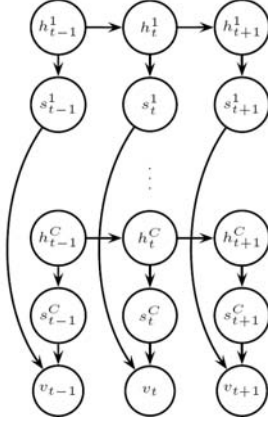

Figure 1: The structure of the LGSSM for ICA.

A particular case for which the Bayesian LGSSM is of interest is in extracting independent source signals underlying a multivariate time-series [5, 15]. This will demonstrate how the approach developed in Section 3 makes VB easily to apply. The sources $s^i$ are modeled as independent in the following sense:

$$p(s^i_{1:T}, s^j_{1:T}) = p(s^i_{1:T})p(s^j_{1:T}), \qquad \text{for } i \neq j, \qquad i, j = 1, \ldots, C.$$

Independence implies block diagonal transition and state noise matrices $A$, $\Sigma_H$ and $\Sigma$, where each block $c$ has dimension $H_c$. A one dimensional source $s^c_t$ for each independent dynamical subsystem is then formed from $s^c_t = \mathbf{1}^\mathsf{T}_c h^c_t$, where $\mathbf{1}_c$ is a unit vector and $h^c_t$ is the state of dynamical system $c$. Combining the sources, we can write $s_t = Ph_t$, where $P = diag(\mathbf{1}^\mathsf{T}_1, \ldots, \mathbf{1}^\mathsf{T}_C)$, $h_t = vert(h^1_t, \ldots, h^C_t)$. The resulting emission matrix is constrained to be of the form $B = WP$, where $W$ is the $V \times C$ mixing matrix. This means that the observations are formed from linearly mixing the sources, $v_t = Ws_t + \eta^v_t$. The graphical structure of this model is presented in Fig 1. To encourage redundant components to be removed, we place a zero mean Gaussian prior on $W$. In this case, we do not define a prior for the parameters $\Sigma_H$ and $\Sigma_V$ which are instead considered as hyperparameters. More details of the model are given in [15]. The constraint $B = WP$ requires a minor modification from Section 3, as we discuss below.

**Inference on** $q(h_{1:T})$

A small modification of the mean + fluctuation decomposition for $B$ occurs, namely:

$$\left\langle (v_t - Bh_t)^\mathsf{T} \Sigma^{-1}_V (v_t - Bh_t) \right\rangle_{q(W)} = (v_t - \langle B \rangle h_t)^\mathsf{T} \Sigma^{-1}_V (v_t - \langle B \rangle h_t) + h^\mathsf{T}_t P^\mathsf{T} S_W Ph_t,$$

where $\langle B \rangle \equiv \langle W \rangle P$ and $S_W = V H^{-1}_W$. The quantities $\langle W \rangle$ and $H_W$ are obtained as in Appendix A.1 with the replacement $h_t \leftarrow Ph_t$. To represent the above as a LGSSM, we augment $v_t$ and $B$ as

$$\tilde{v}_t = vert(v_t, \mathbf{0}_H, \mathbf{0}_C), \qquad \tilde{B} = vert(\langle B \rangle, U_A, U_W P),$$

where $U_W$ is the Cholesky decomposition of $S_W$. The equivalent LGSSM is then completed by specifying $\tilde{A} \equiv \langle A \rangle$, $\tilde{\Sigma}_H \equiv \Sigma_H$, $\tilde{\Sigma}_V \equiv diag(\Sigma_V, I_H, I_C)$, $\tilde{\mu} \equiv \mu$, $\tilde{\Sigma} \equiv \Sigma$, and inference for $q(h_{1:T})$ performed using Algorithm 1. This demonstrates the elegance and unity of the approach in Section 3, since no new algorithm needs to be developed to perform inference, even in this special constrained parameter case.

### 4.1 Demonstration

As a simple demonstration, we used an LGSSM to generate 3 sources $s^c_t$ with random $5 \times 5$ transition matrices $A^c$, $\mu = \mathbf{0}_H$ and $\Sigma \equiv \Sigma_H \equiv I_H$. The sources were mixed into three observations $v_t = Ws_t + \eta^v_t$, for $W$ chosen with elements from a zero mean unit variance Gaussian distribution, and $\Sigma_V = I_V$. We then trained a Bayesian LGSSM with 5 sources and $7 \times 7$ transition matrices $A^c$. To bias the model to find the simplest sources, we used $\hat{A}^c \equiv \mathbf{0}_{H_c, H_c}$ for all sources. In Fig2a and Fig 2b we see the original sources and the noisy observations respectively. In Fig2c we see the estimated sources from our method after convergence of the hyperparameter updates. Two of the 5 sources have been removed, and the remaining three are a reasonable estimation of the original sources. Another possible approach for introducing prior knowledge is to use a Maximum a Posteriori (MAP)

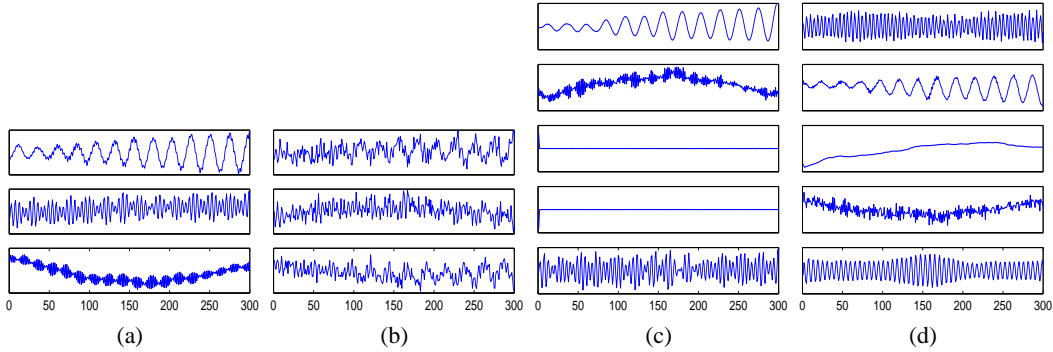

Figure 2: (a) Original sources $s_t$. (b) Observations resulting from mixing the original sources, $v_t = W s_t + \eta_t^v$, $\eta_t^v \sim \mathcal{N}(0, I)$. (c) Recovered sources using the Bayesian LGSSM. (d) Sources found with MAP LGSSM.

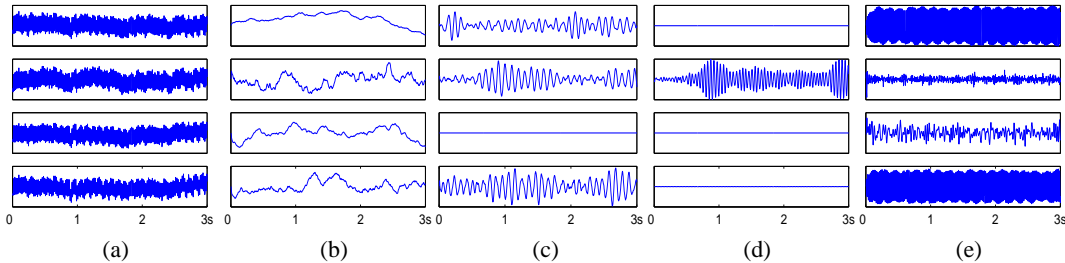

Figure 3: (a) Original raw EEG recordings from 4 channels. (b-e) 16 sources $s_t$ estimated by the Bayesian LGSSM.

procedure by adding a prior term to the original log-likelihood $\log p(v_{1:T}|A, W, \Sigma_H, \Sigma_V, \mu, \Sigma) + \log p(A|\alpha) + \log p(W|\beta)$. However, it is not clear how to reliably find the hyperparameters $\alpha$ and $\beta$ in this case. One solution is to estimate them by optimizing the new objective function jointly with respect to the parameters and hyperparameters (this is the so-called joint map estimation – see for example [16]). A typical result of using this joint MAP approach on the artificial data is presented in Fig 2d. The joint MAP does not estimate the hyperparameters well, and the incorrect number of sources is identified.

## 4.2    Application to EEG Analysis

In Fig 3a we plot three seconds of EEG data recorded from 4 channels (located in the right hemisphere) while a person is performing imagined movement of the right hand. As is typical in EEG, each channel shows drift terms below 1 Hz which correspond to artifacts of the instrumentation, together with the presence of 50 Hz mains contamination and masks the rhythmical activity related to the mental task, mainly centered at 10 and 20 Hz [17]. We would therefore like a method which enables us to extract components in these information-rich 10 and 20 Hz frequency bands. Standard ICA methods such as FastICA do not find satisfactory sources based on raw 'noisy' data, and preprocessing with band-pass filters is usually required. Additionally, in EEG research, flexibility in the number of recovered sources is important since there may be many independent oscillators of interest underlying the observations and we would like some way to automatically determine their effective number. To preferentially find sources at particular frequencies, we specified a block diagonal matrix $\hat{A}^c$ for each source $c$, where each block is a $2 \times 2$ rotation matrix at the desired frequency. We defined the following 16 groups of frequencies: [0.5], [0.5], [0.5], [0.5]; [10,11], [10,11], [10,11], [10,11]; [20,21], [20,21], [20,21], [20,21]; [50], [50], [50], [50]. The temporal evolution of the sources obtained after training the Bayesian LGSSM is given in Fig 3(b,c,d,e) (grouped by frequency range). The Bayes LGSSM removed 4 unnecessary sources from the mixing matrix $W$, that is one [10,11] Hz and three [20,21] Hz sources. The first 4 sources contain dominant low frequency drift, sources 5, 6 and 8 contain [10,11] Hz, while source 10 contains [20,21] Hz centered activity. Of the 4 sources initialized to 50 Hz, only 2 retained 50 Hz activity, while the $A^c$ of the

other two have changed to model other frequencies present in the EEG. This method demonstrates the usefulness and applicability of the VB method in a real-world situation.

## 5   Conclusion

We considered the application of Variational Bayesian learning to Linear Gaussian State-Space Models. This is an important class of models with widespread application, and finding a simple way to implement this approximate Bayesian procedure is of considerable interest. The most demanding part of the procedure is inference of the hidden states of the model. Previously, this has been achieved using Belief Propagation, which differs from inference in the Kalman Filtering/Smoothing literature, for which highly efficient and stabilized procedures exist. A central contribution of this paper is to show how inference *can* be written using the standard Kalman Filtering/Smoothing recursions by augmenting the original model. Additionally, a minor modification to the standard Kalman Filtering routine may be applied for computational efficiency. We demonstrated the elegance and unity of our approach by showing how to easily apply a Variational Bayes analysis of temporal ICA. Specifically, our Bayes ICA approach successfully extracts independent processes underlying EEG signals, biased towards preferred frequency ranges. We hope that this simple and unifying interpretation of Variational Bayesian LGSSMs may therefore facilitate the further application to related models.

## A   Parameter Updates for $A$ and $B$

### A.1   Determining $q(B|\Sigma_V)$

By examining $\mathcal{F}$, the contribution of $q(B|\Sigma_V)$ can be interpreted as the negative KL divergence between $q(B|\Sigma_V)$ and a Gaussian. Hence, optimally, $q(B|\Sigma_V)$ is a Gaussian. The covariance $[\Sigma_B]_{ij,kl} \equiv \left\langle \big(B_{ij} - \langle B_{ij}\rangle\big)\big(B_{kl} - \langle B_{kl}\rangle\big)\right\rangle$ (averages wrt $q(B|\Sigma_V)$) is given by:

$$[\Sigma_B]_{ij,kl} = [H_B^{-1}]_{jl}\,[\Sigma_V]_{ik}\,, \quad \text{where } [H_B]_{jl} \equiv \sum_{t=1}^{T} \left\langle h_t^j h_t^l \right\rangle_{q(h_t)} + \beta_j \delta_{jl}.$$

The mean is given by $\langle B\rangle = N_B H_B^{-1}$, where $[N_B]_{ij} \equiv \sum_{t=1}^{T} \left\langle h_t^j \right\rangle_{q(h_t)} v_t^i + \beta_j \hat{B}_{ij}$.

### Determining $q(A|\Sigma_H)$

Optimally, $q(A|\Sigma_H)$ is a Gaussian with covariance

$$[\Sigma_A]_{ij,kl} = [H_A^{-1}]_{jl}\,[\Sigma_H]_{ik}\,, \quad \text{where } [H_A]_{jl} \equiv \sum_{t=1}^{T-1} \left\langle h_t^j h_t^l \right\rangle_{q(h_t)} + \alpha_j \delta_{jl}.$$

The mean is given by $\langle A\rangle = N_A H_A^{-1}$, where $[N_A]_{ij} \equiv \sum_{t=2}^{T} \left\langle h_{t-1}^j h_t^i \right\rangle_{q(h_{t-1:t})} + \alpha_j \hat{A}_{ij}$.

## B   Covariance Updates

By specifying a Wishart prior for the inverse of the covariances, conjugate update formulae are possible. In practice, it is more common to specify diagonal inverse covariances, for which the corresponding priors are simply Gamma distributions [7, 5]. For this simple diagonal case, the explicit updates are given below.

### Determining $q(\Sigma_V)$

For the constraint $\Sigma_V^{-1} = diag(\rho)$, where each diagonal element follows a Gamma prior $Ga(b_1, b_2)$ [7], $q(\rho)$ factorizes and the optimal updates are

$$q(\rho_i) = Ga\left(b_1 + \frac{T}{2}, b_2 + \frac{1}{2}\left(\sum_{t=1}^{T}(v_t^i)^2 - [G_B]_{ii} + \sum_j \beta_j \hat{B}_{ij}^2\right)\right),$$

where $G_B \equiv N_B H_B^{-1} N_B^{\mathsf{T}}$.

**Determining $q(\Sigma_H)$**

Analogously, for $\Sigma_H^{-1} = diag(\tau)$ with prior $Ga(a_1, a_2)$ [5], the updates are

$$q(\tau_i) = Ga\left(a_1 + \frac{T-1}{2}, a_2 + \frac{1}{2}\left(\sum_{t=2}^{T}\langle (h_t^i)^2\rangle - [G_A]_{ii} + \sum_j \alpha_j \hat{A}_{ij}^2\right)\right),$$

where $G_A \equiv N_A H_A^{-1} N_A^{\mathsf{T}}$.

**Acknowledgments**

This work is supported by the European DIRAC Project FP6-0027787. This paper only reflects the authors' views and funding agencies are not liable for any use that may be made of the information contained herein.

**References**

[1] Y. Bar-Shalom and X.-R. Li. *Estimation and Tracking: Principles, Techniques and Software*. Artech House, 1998.

[2] M. S. Grewal and A. P. Andrews. *Kalman Filtering: Theory and Practice Using MATLAB*. John Wiley and Sons, Inc., 2001.

[3] R. H. Shumway and D. S. Stoffer. *Time Series Analysis and Its Applications*. Springer, 2000.

[4] M. J. Beal, F. Falciani, Z. Ghahramani, C. Rangel, and D. L. Wild. A Bayesian approach to reconstructing genetic regulatory networks with hidden factors. *Bioinformatics*, 21:349–356, 2005.

[5] A. T. Cemgil and S. J. Godsill. Probabilistic phase vocoder and its application to interpolation of missing values in audio signals. In *13th European Signal Processing Conference*, 2005.

[6] H. Valpola and J. Karhunen. An unsupervised ensemble learning method for nonlinear dynamic state-space models. *Neural Computation*, 14:2647–2692, 2002.

[7] M. J. Beal. *Variational Algorithms for Approximate Bayesian Inference*. Ph.D. thesis, Gatsby Computational Neuroscience Unit, University College London, 2003.

[8] M. Davy and S. J. Godsill. Bayesian harmonic models for musical signal analysis (with discussion). In J.O. Bernardo, J.O. Berger, A.P Dawid, and A.F.M. Smith, editors, *Bayesian Statistics VII*. Oxford University Press, 2003.

[9] D. J. C. MacKay. Ensemble learning and evidence maximisation. Unpublished manuscipt: www.variational-bayes.org, 1995.

[10] M. Morf and T. Kailath. Square-root algorithms for least-squares estimation. *IEEE Transactions on Automatic Control*, 20:487–497, 1975.

[11] P. Park and T. Kailath. New square-root smoothing algorithms. *IEEE Transactions on Automatic Control*, 41:727–732, 1996.

[12] E. Niedermeyer and F. Lopes Da Silva. *Electroencephalography: basic principles, clinical applications and related fields*. Lippincott Williams and Wilkins, 1999.

[13] S. Roweis and Z. Ghahramani. A unifying review of linear Gaussian models. *Neural Computation*, 11:305–345, 1999.

[14] M. Verhaegen and P. Van Dooren. Numerical aspects of different Kalman filter implementations. *IEEE Transactions of Automatic Control*, 31:907–917, 1986.

[15] S. Chiappa and D. Barber. Bayesian linear Gaussian state-space models for biosignal decomposition. *Signal Processing Letters*, 14, 2007.

[16] S. S. Saquib, C. A. Bouman, and K. Sauer. ML parameter estimation for Markov random fields with applicationsto Bayesian tomography. *IEEE Transactions on Image Processing*, 7:1029–1044, 1998.

[17] G. Pfurtscheller and F. H. Lopes da Silva. Event-related EEG/MEG synchronization and desynchronization: basic principles. *Clinical Neurophysiology*, pages 1842–1857, 1999.

## Footnotes

[1]Also called Kalman Filters/Smoothers, Linear Dynamical Systems.

[2]$v_{1:T}$ denotes $v_1, \ldots, v_T$.

[3]More general Gaussian priors may be more suitable depending on the application.

[4]For expositional simplicity, we do not put priors on $\mu$ and $\Sigma$.

[5]For simplicity, we keep the parameters of the Gamma priors fixed.

[6]Strictly we should write throughout $q(\cdot|v_{1:T})$. We omit the dependence on $v_{1:T}$ for notational convenience.

[7]For simplicity of exposition, we ignore the first time-point here.

[8]The notation $vert(x_1, \ldots, x_n)$ stands for vertically concatenating the arguments $x_1, \ldots, x_n$.

[9]Strictly, we need a time-dependent emission $\tilde{B}_t = \tilde{B}$, for $t = 1, \ldots, T-1$. For time $T$, $\tilde{B}_T$ has the Cholesky factor $U_A$ replaced by $\mathbf{0}_{H,H}$.

[10]There are several ways of achieving a similar augmentation. We chose this since, in the non-Bayesian limit $U_A = U_B = \mathbf{0}_{H,H}$, no numerical instabilities would be introduced.

[11]Note that, since the augmented LGSSM $\tilde{q}(h_{1:T}|\tilde{v}_{1:T})$ is designed to match the *fully* clamped distribution $q(h_{1:T}|v_{1:T})$, the filtered posterior $\tilde{q}(h_t|\tilde{v}_{1:t})$ does not correspond to $q(h_t|v_{1:t})$.

[12]The computation of the log-likelihood bound does not require any augmentation.
